# A Generic Approach for Identification of Event Related Brain Potentials via a Competitive Neural Network Structure

**Daniel H. Lange**
Department of Electrical Engineering
Technion - IIT
Haifa 32000
Israel
e-mail: lange@turbo.technion.ac.il

**Hava T. Siegelmann**
Department of Industrial Engineering
Technion - IIT
Haifa 32000
Israel
e-mail: iehava@ie.technion.ac.il

**Hillel Pratt**
Evoked Potential Laboratory
Technion - IIT
Haifa 32000
Israel
e-mail: hillel@tx.technion.ac.il

**Gideon F. Inbar**
Department of Electrical Engineering
Technion - IIT
Haifa 32000
Israel
e-mail: inbar@ee.technion.ac.il

## Abstract

We present a novel generic approach to the problem of Event Related Potential identification and classification, based on a competitive Neural Net architecture. The network weights converge to the embedded signal patterns, resulting in the formation of a *matched filter* bank. The network performance is analyzed via a simulation study, exploring identification robustness under low SNR conditions and compared to the expected performance from an information theoretic perspective. The classifier is applied to real event-related potential data recorded during a classic *odd-ball* type paradigm; for the first time, within-session variable signal patterns are automatically identified, dismissing the strong and limiting requirement of *a-priori* stimulus-related selective grouping of the recorded data.

# 1   INTRODUCTION

## 1.1   EVENT RELATED POTENTIALS

Ever since Hans Berger's discovery that the electrical activity of the brain can be measured and recorded via surface electrodes mounted on the scalp, there has been major interest in the relationship between such recordings and brain function. The first recordings were concerned with the spontaneous electrical activity of the brain, appearing in the form of rhythmic voltage oscillations, which later received the term *electroencephalogram* or *EEG*. Subsequently, more recent research has concentrated on time-locked brain activity, related to specific events, external or internal to the subject. This time-locked activity, referred to also as Event Related Potentials (ERP's), is regarded as a manifestation of brain processes related to preparation for or in response to discrete events meaningful to the subject.

The ongoing electrical activity of the brain, the EEG, is comprised of relatively slow fluctuations, in the range of 0.1 - 100 Hz, with magnitudes of 10 - 100 uV. ERP's are characterized by overlapping spectra with the EEG, but with significantly lower magnitudes of 0.1 - 10 uV. The unfavorable Signal to Noise Ratio (SNR) requires filtering of the raw signals to enable analysis of the time-locked signals. The common method used for this purpose is signal averaging, synchronized to repeated occurrences of a specific event. Averaging-based techniques assume a deterministic signal within the averaged session, and thus signal variability can not be modeled unless *a-priori* stimulus- or response-based categorization is available; it is the purpose of this paper to provide an alternative working method to enhance conventional averaging techniques, and thus facilitating identification and analysis of variable brain responses.

## 1.2   COMPETITIVE LEARNING

Competitive learning is a well-known branch of the general unsupervised learning theme. The elementary principles of competitive learning are (Rumelhart & Zipser, 1985): (a) start with a set of units that are all the same except for some randomly distributed parameter which makes each of them respond slightly differently to a set of input patterns, (b) limit the strength of each unit, and (c) allow the units to compete in some way for the right to respond to a given subset of inputs. Applying these three principles yields a learning paradigm where individual units learn to specialize on sets of similar patterns and thus become *feature detectors*. Competitive learning is a mechanism well-suited for regularity detection (Haykin, 1994), where there is a population of input patterns each of which is presented with some probability. The detector is supposed to discover statistically salient features of the input population, without *a-priori* categorization into which the patterns are to be classified. Thus the detector needs to develop its own featural representation of the population of input patterns capturing its most salient features.

## 1.3   PROBLEM STATEMENT

The complicated, generally unknown relationships between the stimulus and its associated brain response, and the extremely low SNR of the brain responses which are practically *masked* by the background brain activity, make the choice of a self organizing structure for post-stimulus epoch analysis most appropriate. The competitive network, having the property that its weights converge to the actual embedded signal patterns while inherently averaging out the additive background EEG, is thus an evident choice.

## 2 THE COMPETITIVE NEURAL NETWORK

### 2.1 THEORY

The common architecture of a competitive learning system appears in Fig. 1. The system consists of a set of hierarchically layered neurons in which each layer is connected via excitatory connections with the following layer. Within a layer, the neurons are divided into sets of inhibitory clusters in which all neurons within a cluster inhibit all other neurons in the cluster, which results in a competition among the neurons to respond to the pattern appearing on the previous layer.

Let $w_{ji}$ denote the synaptic weight connecting input node $i$ to neuron $j$. A neuron learns by shifting synaptic weights from its inactive to active input nodes. If a neuron does not respond to some input pattern, no learning occurs in that neuron. When a single neuron wins the competition, each of its input nodes gives up some proportion of its synaptic weight, which is distributed equally among the active input nodes, fulfilling: $\sum_i w_{ji} = 1$. According to the standard competitive learning rule, for a winning neuron to an input vector $x_i$, the change $\Delta w_{ji}$ is defined by: $\Delta w_{ji} = \eta(x_i - w_{ji})$, where $\eta$ is a learning rate coefficient. The effect of this rule is that the synaptic weights of a winning neuron are shifted towards the input pattern; thus assuming zero-mean additive background EEG, once converged, the network operates as a *matched filter* bank classifier.

### 2.2 MATCHED FILTERING

From an information theoretic perspective, once the network has converged, our classification problem coincides with the general detection problem of known signals in additive noise. For simplicity, we shall limit the discussion to the binary decision problem of a known signal in additive white Gaussian noise, expandable to the M-ary detection in colored noise (Van Trees, 1968).

Adopting the common assumption of EEG and ERP additivity (Gevins, 1984), and distinct signal categories, the competitive NN weights inherently converge to the general signal patterns embedded within the background brain activity; therefore the converged network operates as a *matched filter* bank. Assuming the simplest binary decision problem, the received signal under one hypothesis consists of a completely known signal, $\sqrt{E}s(t)$, representing the EP, corrupted by an additive zero-mean Gaussian noise $w(t)$ with variance $\sigma^2$; the received signal under the other hypothesis consists of the noise $w(t)$ alone. Thus:

$$H_0: \quad r(t) = w(t), \qquad\qquad 0 \le t \le T$$
$$H_1: \quad r(t) = \sqrt{E}s(t) + w(t), \quad 0 \le t \le T$$

For convenience we assume that $\int_0^T s^2(t)dt = 1$, so that $E$ represents the signal energy. The problem is to observe $r(t)$ over the interval $[0,T]$ and decide whether $H_0$ or $H_1$ is true. It can be shown that the *matched filter* is the optimal detector, its impulse response being simply the signal reversed in time and shifted:

$$h(\tau) = s(T - \tau) \tag{1}$$

Assuming that there is no *a-priori* knowledge of the probability of signal presence, the total probability of error depends only on the SNR and is given by (Van Trees, 1968):

$$Pe = \frac{1}{\sqrt{2\pi}} \int_{\sqrt{\frac{E}{\sigma^2}}}^{\infty} \exp(-\frac{x^2}{2})dx \tag{2}$$

Fig. 2 presents the probability of true detection: (a) as a function of SNR, for minimized error probability, and (b) as a function of the probability of false detection. These

results are applicable to our detection problem assuming approximate Gaussian EEG characteristics (Gersch, 1970), or optimally by using a pre-whitening approach (Lange et. al., 1997).

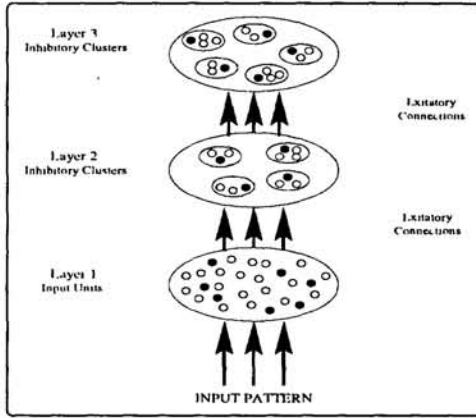

Figure 1: *The architecture of a competitive learning structure: learning takes place in hierarchically layered units, presented as filled (active) and empty (inactive) dots.*

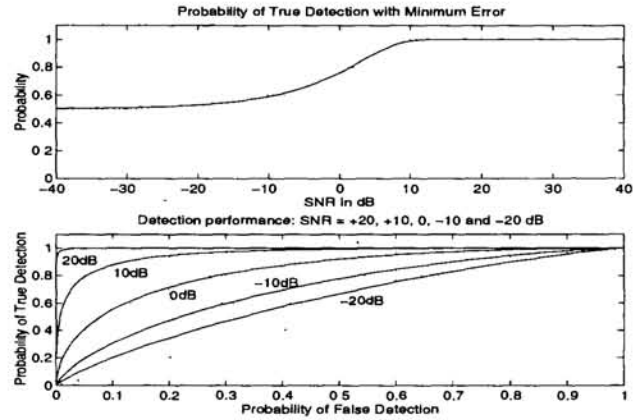

Figure 2: *Detection performance. Top: probability of detection as a function of the SNR. Bottom: detection characteristics.*

## 2.3 NETWORK TRAINING AND CONVERGENCE

Our net includes a 300-node input layer and a competitive layer consisting of single-layered competing neurons. The network weights are initialized with random values and trained with the standard competitive learning rule, applied to the normalized input vectors:

$$\Delta w_{ji} = \eta \left( \frac{x_i}{\sum_i x_i} - w_{ji} \right) \tag{3}$$

The training is applied to the winning neuron of each epoch, while increasing the bias of the frequently winning neuron to gradually reduce its chance of winning consecutively (eliminating the *dead neuron* effect (Freeman & Skapura, 1992)). Symmetrically, its bias is reduced with the winnings of other neurons.

In order to evaluate the network performance, we explore its convergence by analyzing the learning process via the continuously adapting weights:

$$\rho_j(n) = \sqrt{\sum_i \Delta w_{ji}^2} \quad ; \quad j = 1, 2, ..., C \tag{4}$$

where $C$ represents the pre-defined number of categories. We define a set of classification confidence coefficients of the converged network:

$$\Gamma_j = 1 - \frac{\rho_j(N)}{\max_j\{\rho_j(N)\}} \tag{5}$$

Assuming existence of a null category, in which the measurements include only background noise (EEG), $\max_j\{\rho_j(N)\}$ corresponds to the noise variance. Thus the values of $\Gamma_j$, the confidence coefficients, ranging from 0 to 1 (random classification to completely separated categories), indicate the reliability of classification, which breaks down with the fall of SNR. Finally, it should be noted that an explicit statistical evaluation of the network convergence properties can be found in (Lange, 1997).

## 2.4 SIMULATION STUDY

A simulation study was carried out to assess the performance of the competitive network classification system. A moving average (MA) process of order 8 (selected according to Akaike's condition applied to ongoing EEG (Gersch, 1970)), driven by a deterministic realization of a Gaussian white noise series, simulated the ongoing background activity $x(n)$. An average of 40 single-trials from a cognitive odd-ball type experiment (to be explained in the Experimental Study), was used as the signal $s(n)$. Then, five 100-trial ensembles were synthesized, to study the classification performance under variable SNR conditions. A sample realization and its constituents, at an SNR of 0 dB, is shown in Fig. 3. The simulation included embedding the signal $s(n)$ in the synthesized background activity $x(n)$ at five SNR levels (-20,-10,0,+10, and +20 dB), and training the network with 750 sweeps (per SNR level). Fig. 4 shows the convergence patterns and classification confidences of the two neurons, where it can be seen that for SNR's lower than $-10dB$ the classification confidence declines sharply.

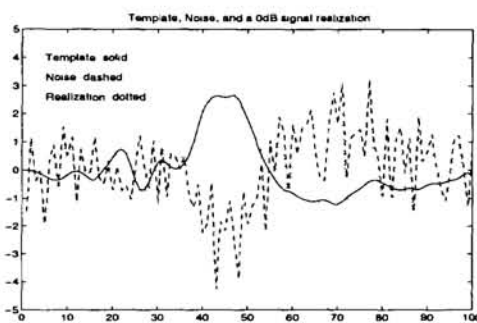

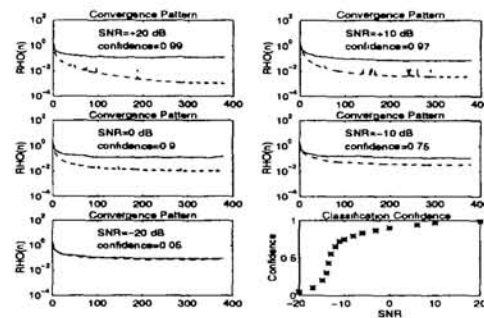

Figure 3: *A sample single realization (dotted) and its constituents (signal - solid, noise - dashed). SNR = 0 dB.*

Figure 4: *Convergence patterns and classification confidence values for varying SNR levels.*

The classification results, tested on 100 input vectors, 50 of each category, for each SNR, are presented in the table below; due to the competitive scheme, Positives and False Negatives as well as Negatives and False Positives are complementary. These empirical results are in agreement with the analytical results presented in the above Matched Filtering section.

Table 1: Classification Results

|            | Pos  | Neg  | FP  | FN  |
|------------|------|------|-----|-----|
| snr=+20dB  | 100% | 100% | 0%  | 0%  |
| snr=+10dB  | 100% | 100% | 0%  | 0%  |
| snr= 0dB   | 100% | 100% | 0%  | 0%  |
| snr=-10dB  | 88%  | 92%  | 8%  | 12% |
| snr=-20dB  | 58%  | 54%  | 46% | 42% |

# 3  EXPERIMENTAL STUDY

## 3.1  MOTIVATION

An important task in ERP research is to identify effects related to cognitive processes triggered by meaningful versus non-relevant stimuli. A common procedure to study these effects is the classic *odd-ball* paradigm, where the subject is exposed to a random

sequence of stimuli and is instructed to respond only to the task-relevant (Target) ones. Typically, the brain responses are extracted via selective averaging of the recorded data, ensembled according to the types of related stimuli. This method of analysis assumes that the brain responds equally to the members of each type of stimulus; however the validity of this assumption is unknown in this case where cognition itself is being studied. Using our proposed approach, *a-priori* grouping of the recorded data is not required, thus overcoming the above severe assumption on cognitive brain function. The results of applying our method are described below.

## 3.2 EXPERIMENTAL PARADIGM

Cognitive event-related potential data was acquired during an odd-ball type paradigm from Pz referenced to the mid-lower jaw, with a sample frequency of 250 Hz (Lange et. al., 1995). The subject was exposed to repeated visual stimuli, consisting of the digits '3' and '5', appearing on a PC screen. The subject was instructed to press a push-button upon the appearance of '5' – the *Target* stimulus, and ignore the appearances of the digit '3'.

With odd-ball type paradigms, the Target stimulus is known to elicit a prominent positive component in the ongoing brain activity, related to the identification of a meaningful stimulus. This component has been labeled $P_{300}$, indicating its polarity (positive) and timing of appearance (300 ms after stimulus presentation). The parameters of the $P_{300}$ component (latency and amplitude) are used by neurophysiologists to assess effects related to the relevance of stimulus and level of attention (Lange et. al., 1995).

## 3.3 IDENTIFICATION RESULTS

The competitive network was trained with 80 input vectors, half of which were Target ERP's and the other half were Non Target. The network converged after approximately 300 iterations (per neuron), yielding a reasonable confidence coefficient of 0.7.

A sample of two single-trial post-stimulus sweeps, of the Target and Non-Target averaged ERP templates and of the NN identified signal categories, are presented in Fig. 5. The convergence pattern is shown in Fig. 6. The automatic identification procedure has provided two signal categories, with almost perfect matches to the stimulus-related selective averaged signals. The obtained categorization confirms the usage of averaging methods for this classic experiment, and thus presents an important result in itself.

## 4   DISCUSSION AND CONCLUSION

A generic system for identification and classification of single-trial ERP's was presented. The simulation study demonstrated the powerful capabilities of the competitive neural net in classifying the low amplitude signals embedded within the large background noise. The detection performance declined rapidly for SNR's lower than $-10dB$, which is in general agreement with the theoretical statistical results, where loss of significance in detection probability is evident for SNR's lower than $-20dB$. Empirically, high classification performance was maintained with SNR's of down to $-10dB$, yielding confidences in the order of 0.7 or higher.

The experimental study presented an unsupervised identification and classification of the raw data into Target and Non-Target responses, dismissing the requirement of stimulus-related selective data grouping. The presented results indicate that the noisy brain responses may be identified and classified objectively in cases where relevance of

the stimuli is unknown or needs to be determined, e.g. in lie-detection scenarios (Lange & Inbar, 1996), and thus open new possibilities in ERP research.

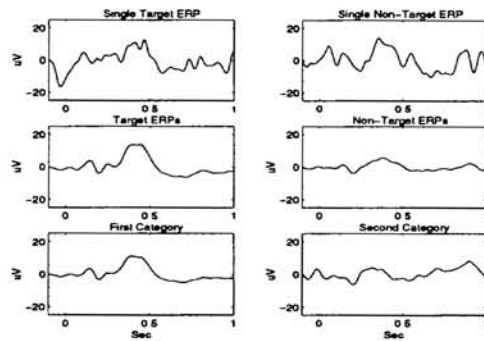

Figure 5: *Top row: sample raw Target and Non-Target sweeps. Middle row: Target and Non-Target ERP templates. Bottom row: the NN categorized patterns.*

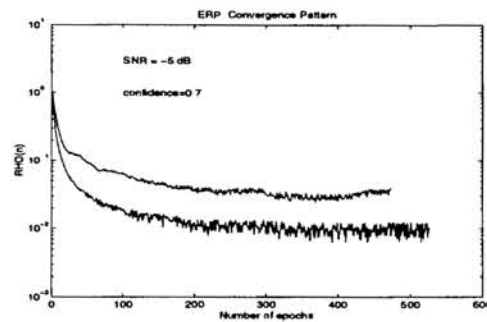

Figure 6: *Convergence pattern of the ERP categorization process; convergence is achieved after 300 iterations per neuron.*

## References

[1] Freeman J.A. and Skapura D.M. *Neural Networks: Algorithms, Applications, and Programming Techniques:* Addison-Wesley Publishing Company, USA, 1992.

[2] Gersch W., "Spectral Analysis of EEG's by Autoregressive Decomposition of Time Series," *Math. Biosc.,* vol. 7, pp. 205-222, 1970.

[3] Gevins A.S., "Analysis of the Electromagnetic Signals of the Human Brain: Milestones, Obstacles, and Goals," *IEEE Trans. Biomed. Eng.,* vol. BME-31, pp. 833-850, 1984.

[4] Haykin S. *Neural Networks: A Comprehensive Foundation.* Macmillan College Publishing Company, Inc., USA, 1994.

[5] Lange D. H. *Modeling and Estimation of Transient, Evoked Brain Potentials.* D.Sc. dissertation, Techion - Israel Institute of Technology, 1997.

[6] Lange D.H. and Inbar G.F., "Brain Wave Based Polygraphy," *Proceedings of the IEEE EMBS96 - the 18th Annual International Conference of the IEEE Engineering on Medicine and Biology Society,* Amsterdam, October 1996.

[7] Lange D.H., Pratt H. and Inbar G.F., "Modeling and Estimation of Single Evoked Brain Potential Components", *IEEE. Trans. Biomed. Eng.,* vol. BME-44, pp. 791-799, 1997.

[8] Lange D.H., Pratt H., and Inbar G.F., "Segmented Matched Filtering of Single Event Related Evoked Potentials," *IEEE. Trans. Biomed. Eng.,* vol. BME-42, pp. 317-321, 1995.

[9] Rumelhart D.E. and Zipser D., "Feature Discovery by Competitive Learning," *Cognitive Science,* vol. 9, pp. 75-112, 1985.

[10] Van Trees H.L. *Detection, Estimation, and Modulation Theory: Part 1:* John Wiley and Sons, Inc., USA, 1968.
